# Improving Policies without Measuring Merits

**Peter Dayan**[1]
CBCL
E25-201, MIT
Cambridge, MA 02139
dayan@ai.mit.edu

**Satinder P Singh**
Harlequin, Inc
1 Cambridge Center
Cambridge, MA 02142
singh@harlequin.com

## Abstract

Performing policy iteration in dynamic programming should only require knowledge of relative rather than absolute measures of the utility of actions (Werbos, 1991) – what Baird (1993) calls the *advantages* of actions at states. Nevertheless, most existing methods in dynamic programming (including Baird's) compute some form of absolute utility function. For smooth problems, advantages satisfy two differential consistency conditions (including the requirement that they be free of curl), and we show that enforcing these can lead to appropriate policy improvement solely in terms of advantages.

## 1 Introduction

In deciding how to change a policy at a state, an agent only needs to know the differences (called advantages) between the total return based on taking each action $a$ for one step and then following the policy forever after, and the total return based on always following the policy (the conventional *value* of the state under the policy). The advantages are like differentials – they do not depend on the local levels of the total return. Indeed, Werbos (1991) defined Dual Heuristic Programming (DHP), using these facts, learning the derivatives of these total returns with respect to the state. For instance, in a conventional undiscounted maze problem with a

penalty for each move, the advantages for the actions might typically be $-1, 0$ or $1$, whereas the values vary between $0$ and the maximum distance to the goal. Advantages should therefore be easier to represent than absolute value functions in a generalising system such as a neural network and, possibly, easier to learn. Although the advantages are differential, existing methods for learning them, notably Baird (1993), require the agent simultaneously to learn the total return from each state. The underlying trouble is that advantages do not appear to satisfy any form of a Bellman equation. Whereas it is clear that the value of a state should be closely related to the value of its neighbours, it is not obvious that the advantage of action $a$ at a state should be equally closely related to its advantages nearby.

In this paper, we show that under some circumstances it is possible to use a solely advantage-based scheme for policy iteration using the spatial derivatives of the value function rather than the value function itself. Advantages satisfy a particular consistency condition, and, given a model of the dynamics and reward structure of the environment, an agent can use this condition to directly acquire the spatial derivatives of the value function. It turns out that the condition alone may not impose enough constraints to specify these derivatives (this is a consequence of the problem described above) – however the value function is like a potential function for these derivatives, and this allows extra constraints to be imposed.

## 2  Continuous DP, Advantages and Curl

Consider the problem of controlling a deterministic system to minimise $V^*(\mathbf{x}_0) = \min_{\mathbf{u}(t)} \int_0^\infty r(\mathbf{y}(t), \mathbf{u}(t)) dt$, where $\mathbf{y}(t) \in \Re^n$ is the state at time $t$, $\mathbf{u}(t) \in \Re^m$ is the control, $\mathbf{y}(0) = \mathbf{x}_0$, and $\dot{\mathbf{y}}(t) = \mathbf{f}((\mathbf{y}(t), \mathbf{u}(t))$. This is a simplified form of a classic variational problem since $r$ and $\mathbf{f}$ do not depend on time $t$ explicitly, but only through $\mathbf{y}(t)$ and there are no stopping time or terminal conditions on $\mathbf{y}(t)$ (see Peterson, 1993; Atkeson, 1994, for recent methods for solving such problems). This means that the optimal $\mathbf{u}(t)$ can be written as a function of $\mathbf{y}(t)$ and that $V(\mathbf{x}_0)$ is a function of $\mathbf{x}_0$ and not $t$. We do not treat the cases in which the infinite integrals do not converge comfortably and we will also assume adequate continuity and differentiability.

**The solution by advantages:** This problem can be solved by writing down the Hamilton-Jacobi-Bellman (HJB) equation (see Dreyfus, 1965) which $V^*(\mathbf{x})$ satisfies:

$$0 = \min_{\mathbf{u}} \left[ r(\mathbf{x}, \mathbf{u}) + \mathbf{f}(\mathbf{x}, \mathbf{u}) \cdot \nabla_{\mathbf{x}} V^*(\mathbf{x}) \right] \qquad (1)$$

This is the continuous space/time analogue of the conventional Bellman equation (Bellman, 1957) for discrete, non-discounted, deterministic decision problems, which says that for the optimal value function $V^*$, $0 = \min_a \left[ r(x, a) + V^*(f(x, a)) - V^*(x) \right]$, where starting the process at state $x$ and using action $a$ incurs a cost $r(x, a)$ and leaves the process in state $f(x, a)$. This, and its obvious stochastic extension to Markov decision processes, lie at the heart of temporal difference methods for reinforcement learning (Sutton, 1988; Barto, Sutton & Watkins, 1989; Watkins, 1989). Equation 1 describes what the *optimal* value function must satisfy. Discrete dynamic programming also comes with a method called value iteration which starts with any function $V_0(\mathbf{x})$, improves it sequentially, and converges to the optimum.

The alternative method, policy iteration (Howard, 1960), operates in the space of

policies, *ie* functions $\mathbf{w}(\mathbf{x})$. Starting with $\mathbf{w}(\mathbf{x})$, the method requires evaluating everywhere the value function $V^{\mathbf{w}}(\mathbf{x}) = \int_0^\infty r(\mathbf{y}(t), \mathbf{w}(\mathbf{y}(t))dt$, where $\mathbf{y}(0) = \mathbf{x}$, and $\dot{\mathbf{y}}(t) = \mathbf{f}(\mathbf{y}(t), \mathbf{w}(\mathbf{y}(t)))$. It turns out that $V^{\mathbf{w}}$ satisfies a close relative of equation 1:

$$0 = r(\mathbf{x}, \mathbf{w}(\mathbf{x})) + \mathbf{f}(\mathbf{x}, \mathbf{w}(\mathbf{x})) \cdot \nabla_{\mathbf{x}} V^{\mathbf{w}}(\mathbf{x}) \qquad (2)$$

In policy iteration, $\mathbf{w}(\mathbf{x})$ is improved, by choosing the maximising action:

$$\mathbf{w}'(\mathbf{x}) = \mathrm{argmax}_{\mathbf{u}} \left[ r(\mathbf{x}, \mathbf{u}) + \mathbf{f}(\mathbf{x}, \mathbf{u}) \cdot \nabla_{\mathbf{x}} V^{\mathbf{w}}(\mathbf{x}) \right] \qquad (3)$$

as the new action. For discrete Markov decision problems, the equivalent of this process of policy improvement is guaranteed to improve upon $\mathbf{w}$.

In the discrete case and for an analogue of value iteration, Baird (1993) defined the optimal advantage function $A^*(x, a) = \left[ Q^*(x, a) - \max_b Q^*(x, b) \right] / \delta t$, where $\delta t$ is effectively a characteristic time for the process which was taken to be 1 above, and the optimal $Q$ function (Watkins, 1989) is $Q^*(x, a) = r(x, a) + V^*(f(x, a))$, where $V^*(y) = \max_b Q^*(y, b)$. It turns out (Baird, 1993) that in the discrete case, one can cast the whole of policy iteration in terms of advantages. In the continuous case, we define advantages directly as

$$A^{\mathbf{w}}(\mathbf{x}, \mathbf{u}) = r(\mathbf{x}, \mathbf{u}) + \mathbf{f}(\mathbf{x}, \mathbf{u}) \cdot \nabla_{\mathbf{x}} V^{\mathbf{w}}(\mathbf{x}) \qquad (4)$$

This equation indicates how the spatial derivatives of $V^{\mathbf{w}}$ determine the advantages. Note that the consistency condition in equation 2 can be written as $A^{\mathbf{w}}(\mathbf{x}, \mathbf{w}(\mathbf{x})) = 0$. Policy iteration can proceed using

$$\mathbf{w}'(\mathbf{x}) = \mathrm{argmax}_{\mathbf{u}} A^{\mathbf{w}}(\mathbf{x}, \mathbf{u}). \qquad (5)$$

**Doing without $V^{\mathbf{w}}$:** We can now state more precisely the intent of this paper: a) the consistency condition in equation 2 provides constraints on the spatial derivatives $\nabla_{\mathbf{x}} V^{\mathbf{w}}(\mathbf{x})$, at least given a model of $r$ and $\mathbf{f}$; b) equation 4 indicates how these spatial derivatives can be used to determine the advantages, again using a model; and c) equation 5 shows that the advantages *tout court* can be used to improve the policy. Therefore, one apparently should have no need to know $V^{\mathbf{w}}(x)$ but just its spatial derivatives in order to do policy iteration.

**Didactic Example — LQR:** To make the discussion more concrete, consider the case of a one-dimensional linear quadratic regulator (LQR). The task is to minimise $V^*(x_0) = \int_0^\infty \alpha x(t)^2 + \beta u(t)^2 dt$ by choosing $u(t)$, where $\alpha, \beta > 0, \dot{x}(t) = -[ax(t) + u(t)]$ and $x(0) = x_0$. It is well known (*eg* Athans & Falb, 1966) that the solution to this problem is that $V^*(x) = k^* x^2 / 2$ where $k^* = (\alpha + \beta(u^*)^2)/(a + u^*)$ and $u(t) = (-a + \sqrt{a^2 + \alpha/\beta})x(t)$. Knowing the form of the problem, we consider policies $w$ that make $u(t) = wx(t)$ and require $h(x, k) \equiv \nabla_x V^w(x) = kx$, where the correct value of $k = (\alpha + \beta w^2)/(a + w)$. The consistency condition in equation 2 evaluated at state $x$ implies that $0 = (\alpha + \beta w^2)x^2 - h(x, k)(a + w)x$. Doing online gradient descent in the square inconsistency at samples $x_n$ gives $k_{n+1} = k_n - \epsilon \partial \left[ (\alpha + \beta w^2)x_n^2 - k_n x_n(a + w)x_n \right]^2 / \partial k_n$, which will reduce the square inconsistency for small enough $\epsilon$ unless $x = 0$. As required, the square inconsistency can only be zero for all values of $x$ if $k = (\alpha + \beta w^2)/((a + w))$. The advantage of performing action $v$ (note this is *not* $vx$) at state $x$ is, from equation 4, $A^w(x, v) = \alpha x^2 + \beta v^2 - (ax + v)(\alpha + \beta w^2)x/(a + w)$, which, minimising over $v$ (equation 5) gives $u(x) = w'x$ where $w' = (\alpha + \beta w^2)/(2\beta(a + w))$, which is the Newton-Raphson iteration to solve the quadratic equation that determines the optimal policy. In this case, without ever explicitly forming $V^w(x)$, we have been able to learn an optimal

policy. This was based, at least conceptually, on samples $x_n$ from the interaction of the agent with the world.

**The curl condition:** The astute reader will have noticed a problem. The consistency condition in equation 2 constrains the spatial derivatives $\nabla_x V^{\mathbf{w}}$ in only one direction at every point – along the route $\mathbf{f}(\mathbf{x}, \mathbf{w}(\mathbf{x}))$ taken according to the policy there. However, in evaluating actions by evaluating their advantages, we need to know $\nabla_x V^{\mathbf{w}}$ in all the directions accessible through $\mathbf{f}(\mathbf{x}, \mathbf{u})$ at state $\mathbf{x}$. The quadratic regulation task was only solved *because* we employed a function approximator (which was linear in this case $h(x, k) = kx$). For the case of LQR, the restriction that $\mathbf{h}$ be linear allowed information about $\mathbf{f}(\mathbf{x}', \mathbf{w}(\mathbf{x}')) \cdot \nabla_{x'} V^{\mathbf{w}}(\mathbf{x}')$ at distant states $\mathbf{x}'$ and for the policy actions $\mathbf{w}(\mathbf{x}')$ there to determine $\mathbf{f}(\mathbf{x}, \mathbf{u}) \cdot \nabla_x V^{\mathbf{w}}(\mathbf{x})$ at state $\mathbf{x}$ but for non-policy actions $\mathbf{u}$. If we had tried to represent $h(x, k)$ using a more flexible approximator such as radial basis functions, it might not have worked. In general, if we didn't know the form of $\nabla_x V^{\mathbf{w}}(\mathbf{x})$, we cannot rely on the function approximator to generalize correctly.

There is one piece of information that we have yet to use – function $\mathbf{h}(\mathbf{x}, \mathbf{k}) \equiv \nabla_x V^{\mathbf{w}}(\mathbf{x})$ (with parameters $\mathbf{k}$, and in general non-linear) is the gradient of something – it represents a conservative vector field. Therefore its curl should vanish ($\nabla_x \times \mathbf{h}(\mathbf{x}, \mathbf{k}) = 0$). Two ways to try to satisfy this are to represent $\mathbf{h}$ as a suitably weighted combination of functions that satisfy this condition or to use its square as an additional error during the process of setting the parameters $\mathbf{k}$. Even in the case of the LQR, but in more than one dimension, it turns out to be essential to use the curl condition. For the multi-dimensional case we know that $V^{\mathbf{w}}(\mathbf{x}) = \mathbf{x}^T K^{\mathbf{w}} \mathbf{x}/2$ for some symmetric matrix $K^{\mathbf{w}}$, but enforcing zero curl is the only way to enforce this symmetry.

The curl condition says that knowing how some component of $\nabla_x V^{\mathbf{w}}(\mathbf{x})$ *changes* in some direction (*eg* $\partial \nabla_x V^{\mathbf{w}}(\mathbf{x})_2 / \partial x_1$) does provide information about how some other component *changes* in a different direction (*eg* $\partial \nabla_x V^{\mathbf{w}}(\mathbf{x})_1 / \partial x_2$). This information is only useful up to constants of integration, and smoothness conditions will be necessary to apply it.

## 3  Simulations

We tested the method of approximating $\mathbf{h}^{\mathbf{w}}(\mathbf{x}) = \nabla_x V^{\mathbf{w}}(\mathbf{x})$ as a linearly weighted combination of local conservative vector fields $\mathbf{h}^{\mathbf{w}}(\mathbf{x}) = \sum_{i=1}^{n} c_i^{\mathbf{w}} \nabla_x \phi(\mathbf{x}, \mathbf{z}_i)$, where $c_i^{\mathbf{w}}$ are the approximation weights that are set by enforcing equation 2, and $\phi(\mathbf{x}, \mathbf{z}_i) = e^{-\alpha |\mathbf{x} - \mathbf{z}_i|^2}$ are standard radial basis functions (Broomhead & Lowe, 1988; Poggio & Girosi, 1990). We enforced this condition at a discrete set $\{\mathbf{x}_k\}$ of 100 points scattered in the state space, using as a policy, explicit vectors $\mathbf{u}_k$ at those locations, and employed 49 similarly scattered centres $\mathbf{z}_i$. Issues of learning to approximate conservative and non-conservative vector fields using such sums have been discussed by Mussa-Ivaldi (1992). One advantage of using this representation is that $\psi(\mathbf{x}) = \sum_{i=1}^{n} c_i^{\mathbf{w}} \phi(\mathbf{x}, \mathbf{z}_i)$ can be seen as the system's effective policy evaluation function $V^{\mathbf{w}}(\mathbf{x})$, at least modulo an arbitrary constant (we call this an un-normalised value function).

We chose two 2-dimensional problems to prove that the system works. They share the same dynamics $\dot{x}(t) = -\mathbf{x}(t) + \mathbf{u}(t)$, but have different cost functions:

$$r_{\text{LQR}}(\mathbf{x}(t), \mathbf{u}(t)) = 5|\mathbf{x}(t)|^2 + |\mathbf{u}(t)|^2 \quad , \quad r_{\text{SP}}(\mathbf{x}(t), \mathbf{u}(t)) = |\mathbf{x}(t)|^2 + \sqrt{1 + |\mathbf{u}(t)|^2}$$

(6)

$r_{\text{LQR}}$ makes for a standard linear quadratic regulation problem, which has a quadratic optimal value function and a linear optimal controller as before (although now we are using limited range basis functions instead of using the more appropriate linear form). $r_{\text{SP}}$ has a mixture of a quadratic term in $\mathbf{x}(t)$, which encourages the state to move towards the origin, and a more nearly linear cost term in $\mathbf{u}(t)$, which would tend to encourage a constant speed. All the sample points $\mathbf{x}_k$ and radial basis function centres $\mathbf{z}_i$ were selected within the $\{-1, 1\}^2$ square. We started from a randomly chosen policy with both components of $\mathbf{u}_k$ being samples from the uniform distribution $\mathcal{U}(-.25, .25)$. This was chosen so that the overall dynamics of the system, including the $-\mathbf{x}(t)$ component should lead the agent towards the origin.

Figure 1a shows the initial values of $\mathbf{u}_k$ in the regulator case, where the circles are at the leading edges of the local policies which point in the directions shown with relative magnitudes given by the length of the lines, and (for scale) the central object is the square $\{-0.1, 0.1\}^2$. The 'policy' lines are centred at the 100 $\mathbf{x}_k$ points. Using the basis function representation, equation 2 is an over-determined linear system, and so, the standard Moore-Penrose pseudo-inverse was used to find an approximate solution. The un-normalised approximate value function corresponding to this policy is shown in figure 1b. Its bowl-like character is a feature of the optimal value function. For the LQR case, it is straightforward to perform the optimisation in equation 5 analytically, using the values for $\mathbf{h}^{\mathbf{w}}(\mathbf{x}_k)$ determined by the $c_i^{\mathbf{w}}$. Figure 1c,d show the policy and its associated un-normalised value function after 4 iterations. By this point, the policy and value functions are essentially optimal – the policy shows the agent moves inwards from all $\mathbf{x}_k$ and the magnitudes are linearly related to the distances from the centre. Figure 1e,f show the same at the end point for $r_{\text{SP}}$. One major difference is that we performed the optimisation in equation 5 over a discrete set of values for $\mathbf{u}_k$ rather than analytically. The tendency for the agent to maintain a constant speed is apparent except right near the origin. The bowl is not centred exactly at $(0, 0)$ – which is an approximation error.

## 4  Discussion

This paper has addressed the question of whether it is possible to perform policy iteration using just differential quantities like advantages. We showed that using a conventional consistency condition and a curl constraint on the spatial derivatives of the value function it is possible to learn enough about the value function for a policy to improve upon that policy. Generalisation can be key to the whole scheme. We showed this working on an LQR problem and a more challenging non-LQR case. We only treated 'smooth' problems – addressing discontinuities in the value function, which imply undifferentiability, is clearly key. Care must be taken in interpreting this result. The most challenging problem is the error metric for the approximation. The consistency condition may either under-specify or over-specify the parameters. In the former case, just as for standard approximation theory, one needs prior information to regularise the gradient surface. For many problems there may be spatial discontinuities in the policy evaluation, and therefore this is particularly difficult. If the parameters are over-specified (and, for good generalisation, one would generally be working in this regime), we need to evaluate inconsistencies. Inconsistencies cost exactly to the degree that the optimisation in equation 5 is compromised – but this is impossible to quantify. Note that this problem is not

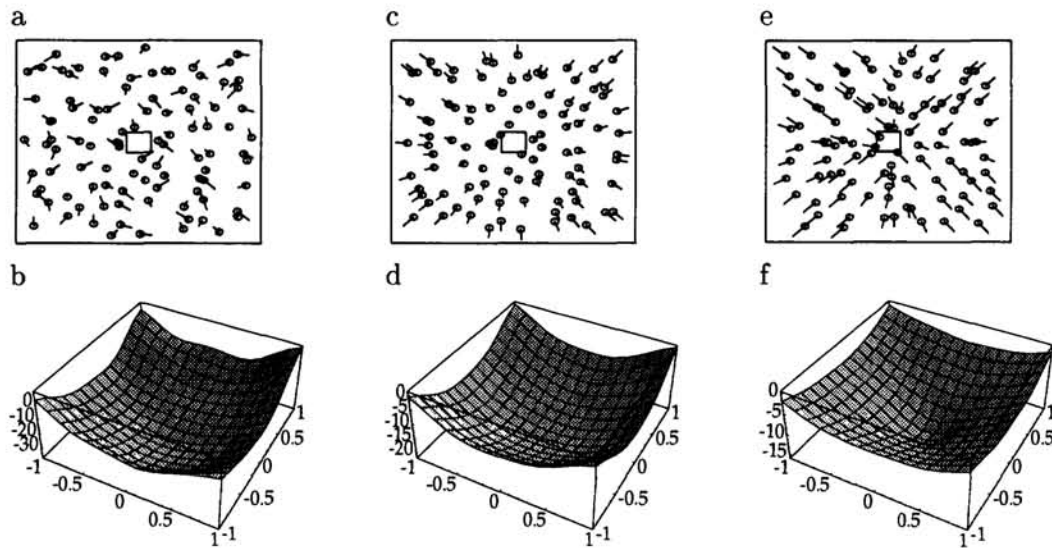

Figure 1: a-d) Policies and un-normalised value functions for the $r_{\text{LQR}}$ and e-f) for the $r_{\text{SP}}$ problem.

confined to the current scheme of learning the derivatives of the value function – it also impacts algorithms based on learning the value function itself. It is also unreasonable to specify the actions $\mathbf{u}_k$ only at the points $\mathbf{x}_k$. In general, one would either need a parameterised function for $\mathbf{u}(\mathbf{x})$ whose parameters would be updated in the light of performing the optimisations in equation 5 (or some sort of interpolation scheme), or alternatively one could generate $\mathbf{u}$ on the fly using the learned values of $\mathbf{h}(\mathbf{x})$.

If there is a discount factor, *ie* $V^*(\mathbf{x}_0) = \min_{\mathbf{u}(t)} \int_0^\infty e^{-\lambda t} r(\mathbf{y}(t), \mathbf{u}(t)) dt$, then $0 = r(\mathbf{x}, \mathbf{w}(\mathbf{x})) - \lambda V^{\mathbf{w}}(\mathbf{x}) + \mathbf{f}(\mathbf{x}, \mathbf{w}(\mathbf{x})) \cdot \nabla_{\mathbf{x}} V^{\mathbf{w}}(\mathbf{x})$ is the equivalent consistency condition to equation 2 (see also Baird, 1993) and so it is no longer possible to learn $\nabla_{\mathbf{x}} V^{\mathbf{w}}(\mathbf{x})$ without ever considering $V^{\mathbf{w}}(\mathbf{x})$ itself. One can still optimise parameterised forms for $V^{\mathbf{w}}$ as in section 3, except that the once arbitrary constant is no longer free.

The discrete analogue to the differential consistency condition in equation 2 amounts to the tautology that given current policy $\pi$, $\forall x$, $A^\pi(x, \pi(x)) = 0$. As in the continuous case, this only provides information about $V^\pi(f(x, \pi(x))) - V^\pi(x)$ and not $V^\pi(f(x, a)) - V^\pi(x)$ for other actions $a$ which are needed for policy improvement. There is an equivalent to the curl condition: if there is a cycle in the undirected transition graph, then the weighted sum of the advantages for the actions along the cycle is equal to the equivalently weighted sum of payoffs along the cycle, where the weights are $+1$ if the action respects the cycle and $-1$ otherwise. This gives a consistency condition that $A^\pi$ has to satisfy – and, just as in the constants of integration for the differential case, it requires grounding: $A^\pi(z, a) = 0$ for some $z$ in the cycle. It is certainly not true that all discrete problems will have sufficient cycles to specify $A^\pi$ completely – in an extreme case, the undirected version of the directed transition graphs might contain no cycles at all. In the continuous case, if the updates are sufficiently smooth, this is not possible. For stochastic problems, the consistency condition equivalent to equation 2 will involve an integral, which,

if doable, would permit the application of our method.

Werbos's (1991) DHP and Mitchell and Thrun's (1993) explanation-based $Q$-learning also study differential forms of the Bellman equation based on differentiating the discrete Bellman equation (or its $Q$-function equivalent) with respect to the state. This is certainly fine as an *additional* constraint that $V^*$ or $Q^*$ must satisfy (as used by Mitchell and Thrun and Werbos' Globalized version of DHP), but by itself, it does not enforce the curl condition, and is insufficient for the whole of policy improvement.

**References**

Athans, M & Falb, PL (1966). *Optimal Control.* New York, NY: McGraw-Hill.

Atkeson, CG (1994). Using Local Trajectory Optimizers To Speed Up Global Optimization in Dynamic Programming. In *NIPS 6.*

Baird, LC, IIIrd (1993). *Advantage Updating.* Technical report, Wright Laboratory, Wright-Patterson Air Force Base.

Barto, AG, Bradtke, SJ & Singh, SP (1995). Learning to act using real-time dynamic programming. *Artificial Intelligence,* **72**, 81-138.

Barto, AG, Sutton, RS & Watkins, CJCH (1990). Learning and sequential decision making. In M Gabriel & J Moore, editors, *Learning and Computational Neuroscience: Foundations of Adaptive Networks.* Cambridge, MA: MIT Press, Bradford Books.

Bellman, RE (1957). *Dynamic Programming.* Princeton, NJ: Princeton University Press.

Broomhead, DS & Lowe, D (1988). Multivariable functional interpolation and adaptive networks. *Complex Systems,* **2**, 321-55.

Dreyfus, SE (1965). *Dynamic Programming and the Calculus of Variations.* New York, NY: Academic Press.

Howard, RA (1960). *Dynamic Programming and Markov Processes.* New York, NY: Technology Press & Wiley.

Mitchell, TM & Thrun, SB (1993). Explanation-based neural network learning for robot control. In *NIPS 5.*

Mussa-Ivaldi, FA (1992). From basis functions to basis fields: Vector field approximation from sparse data. *Biological Cybernetics,* **67**, 479-489.

Peterson, JK (1993). On-Line estimation of optimal value functions. In *NIPS 5.*

Poggio, T & Girosi, F (1990). A theory of networks for learning. *Science,* **247**, 978-982.

Sutton, RS (1988). Learning to predict by the methods of temporal difference. *Machine Learning,* **3**, pp 9-44.

Watkins, CJCH (1989). *Learning from Delayed Rewards.* PhD Thesis. University of Cambridge, England.

Werbos, P (1991). A menu of designs for reinforcement learning over time. In WT Miller IIIrd, RS Sutton & P Werbos, editors, *Neural Networks for Control.* Cambridge, MA: MIT Press, 67-96.

## Footnotes

[1]We are grateful to Larry Saul, Tommi Jaakkola and Mike Jordan for comments, and Andy Barto for pointing out the connection to Werbos' DHP. This work was supported by NSERC, MIT, and grants to Professor Michael I Jordan from ATR Human Information Processing Research and Siemens Corporation.
